# Factorial Hidden Markov Models

**Zoubin Ghahramani**
zoubin@psyche.mit.edu
Department of Computer Science
University of Toronto
Toronto, ON M5S 1A4
Canada

**Michael I. Jordan**
jordan@psyche.mit.edu
Department of Brain & Cognitive Sciences
Massachusetts Institute of Technology
Cambridge, MA 02139
USA

## Abstract

We present a framework for learning in hidden Markov models with distributed state representations. Within this framework, we derive a learning algorithm based on the Expectation–Maximization (EM) procedure for maximum likelihood estimation. Analogous to the standard Baum-Welch update rules, the M-step of our algorithm is exact and can be solved analytically. However, due to the combinatorial nature of the hidden state representation, the exact E-step is intractable. A simple and tractable mean field approximation is derived. Empirical results on a set of problems suggest that both the mean field approximation and Gibbs sampling are viable alternatives to the computationally expensive exact algorithm.

## 1 Introduction

A problem of fundamental interest to machine learning is time series modeling. Due to the simplicity and efficiency of its parameter estimation algorithm, the hidden Markov model (HMM) has emerged as one of the basic statistical tools for modeling discrete time series, finding widespread application in the areas of speech recognition (Rabiner and Juang, 1986) and computational molecular biology (Baldi et al., 1994). An HMM is essentially a mixture model, encoding information about the history of a time series in the value of a single multinomial variable (the hidden state). This multinomial assumption allows an efficient parameter estimation algorithm to be derived (the Baum-Welch algorithm). However, it also severely limits the representational capacity of HMMs. For example, to represent 30 bits of information about the history of a time sequence, an HMM would need $2^{30}$ distinct states. On the other hand an HMM with a *distributed* state representation could achieve the same task with 30 binary units (Williams and Hinton, 1991). This paper addresses the problem of deriving efficient learning algorithms for hidden Markov models with distributed state representations.

The need for distributed state representations in HMMs can be motivated in two ways. First, such representations allow the state space to be decomposed into features that naturally decouple the dynamics of a single process generating the time series. Second, distributed state representations simplify the task of modeling time series generated by the interaction of multiple independent processes. For example, a speech signal generated by the superposition of multiple simultaneous speakers can be potentially modeled with such an architecture.

Williams and Hinton (1991) first formulated the problem of learning in HMMs with distributed state representation and proposed a solution based on deterministic Boltzmann learning. The approach presented in this paper is similar to Williams and Hinton's in that it is also based on a statistical mechanical formulation of hidden Markov models. However, our learning algorithm is quite different in that it makes use of the special structure of HMMs with distributed state representation, resulting in a more efficient learning procedure. Anticipating the results in section 2, this learning algorithm both obviates the need for the two-phase procedure of Boltzmann machines, and has an exact M-step. A different approach comes from Saul and Jordan (1995), who derived a set of rules for computing the gradients required for learning in HMMs with distributed state spaces. However, their methods can only be applied to a limited class of architectures.

## 2  Factorial hidden Markov models

Hidden Markov models are a generalization of mixture models. At any time step, the probability density over the observables defined by an HMM is a mixture of the densities defined by each state in the underlying Markov model. Temporal dependencies are introduced by specifying that the prior probability of the state at time $t$ depends on the state at time $t-1$ through a transition matrix, $P$ (Figure 1a).

Another generalization of mixture models, the cooperative vector quantizer (CVQ; Hinton and Zemel, 1994 ), provides a natural formalism for distributed state representations in HMMs. Whereas in simple mixture models each data point must be accounted for by a single mixture component, in CVQs each data point is accounted for by the combination of contributions from many mixture components, one from each separate vector quantizer. The total probability density modeled by a CVQ is also a mixture model; however this mixture density is assumed to factorize into a product of densities, each density associated with one of the vector quantizers. Thus, the CVQ is a mixture model with distributed representations for the mixture components.

Factorial hidden Markov models[1] combine the state transition structure of HMMs with the distributed representations of CVQs (Figure 1b). Each of the $d$ underlying Markov models has a discrete state $s_i^t$ at time $t$ and transition probability matrix $P_i$. As in the CVQ, the states are mutually exclusive within each vector quantizer and we assume real-valued outputs. The sequence of observable output vectors is generated from a normal distribution with mean given by the weighted combination of the states of the underlying Markov models:

$$\mathbf{y}^t \sim N\left(\sum_{i=1}^{d} W_i \mathbf{s}_i^t, C\right),$$

where $C$ is a common covariance matrix. The $k$-valued states $s_i$ are represented as

discrete column vectors with a 1 in one position and 0 everywhere else; the mean of the observable is therefore a combination of columns from each of the $W_i$ matrices.

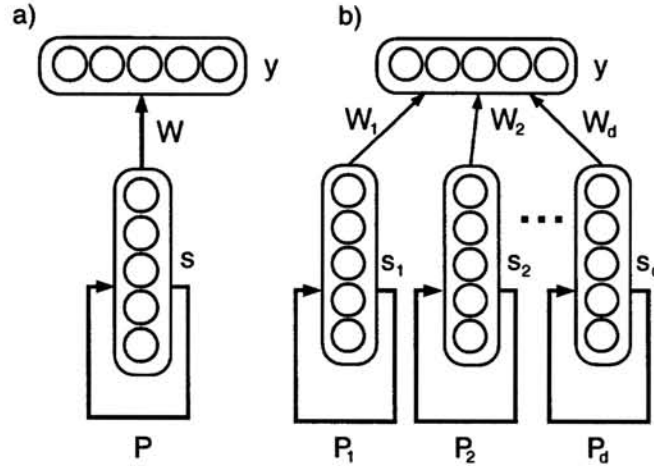

Figure 1. a) Hidden Markov model. b) Factorial hidden Markov model.

We capture the above probability model by defining the energy of a sequence of $T$ states and observations, $\{(\mathbf{s}^t, \mathbf{y}^t)\}_{t=1}^T$, which we abbreviate to $\{\mathbf{s}, \mathbf{y}\}$, as:

$$\mathcal{H}(\{\mathbf{s}, \mathbf{y}\}) = \frac{1}{2} \sum_{t=1}^T \left[\mathbf{y}^t - \sum_{i=1}^d W_i \mathbf{s}_i^t\right]' C^{-1} \left[\mathbf{y}^t - \sum_{i=1}^d W_i \mathbf{s}_i^t\right] - \sum_{t=1}^T \sum_{i=1}^d \mathbf{s}_i^{t\,'} A_i \mathbf{s}_i^{t-1}, \quad (1)$$

where $[A_i]_{jl} = \log P(s_{ij}^t | s_{il}^{t-1})$ such that $\sum_{j=1}^k e^{[A_i]_{jl}} = 1$, and $'$ denotes matrix transpose. Priors for the initial state, $\mathbf{s}^1$, are introduced by setting the second term in (1) to $-\sum_{i=1}^d \mathbf{s}_i^{1\,'} \log \boldsymbol{\pi}_i$. The probability model is defined from this energy by the Boltzmann distribution

$$P(\{\mathbf{s}, \mathbf{y}\}) = \frac{1}{Z} \exp\{-\mathcal{H}(\{\mathbf{s}, \mathbf{y}\})\}. \quad (2)$$

Note that like in the CVQ (Ghahramani, 1995), the unclamped partition function

$$Z = \int d\{\mathbf{y}\} \sum_{\{\mathbf{s}\}} \exp\{-\mathcal{H}(\{\mathbf{s}, \mathbf{y}\})\},$$

evaluates to a constant, independent of the parameters. This can be shown by first integrating the Gaussian variables, removing all dependency on $\{\mathbf{y}\}$, and then summing over the states using the constraint on $e^{[A_i]_{jl}}$.

**The EM algorithm for Factorial HMMs**

As in HMMs, the parameters of a factorial HMM can be estimated via the EM (Baum-Welch) algorithm. This procedure iterates between assuming the current parameters to compute probabilities over the hidden states (E-step), and using these probabilities to maximize the expected log likelihood of the parameters (M-step).

Using the likelihood (2), the expected log likelihood of the parameters is

$$Q(\phi^{\text{new}} | \phi) = \langle -\mathcal{H}(\{\mathbf{s}, \mathbf{y}\}) - \log Z \rangle_c, \quad (3)$$

where $\phi = \{W_i, P_i, C\}_{i=1}^{d}$ denotes the current parameters, and $\langle \cdot \rangle_c$ denotes expectation given the clamped observation sequence and $\phi$. Given the observation sequence, the only random variables are the hidden states. Expanding equation (3) and limiting the expectation to these random variables we find that the statistics that need to be computed for the E-step are $\langle s_i^t \rangle_c$, $\langle s_i^t s_j^{t'} \rangle_c$, and $\langle s_i^t s_i^{t-1'} \rangle_c$. Note that in standard HMM notation (Rabiner and Juang, 1986), $\langle s_i^t \rangle_c$ corresponds to $\gamma_t$ and $\langle s_i^t s_i^{t-1'} \rangle_c$ corresponds to $\xi_t$, whereas $\langle s_i^t s_j^{t'} \rangle_c$ has no analogue when there is only a single underlying Markov model. The M-step uses these expectations to maximize $Q$ with respect to the parameters.

The constant partition function allowed us to drop the second term in (3). Therefore, unlike the Boltzmann machine, the expected log likelihood does not depend on statistics collected in an unclamped phase of learning, resulting in much faster learning than the traditional Boltzmann machine (Neal, 1992).

## M-step

Setting the derivatives of $Q$ with respect to the output weights to zero, we obtain a linear system of equations for $W$:

$$W^{\text{new}} = \left[\sum_{N,t} \langle \mathbf{ss}' \rangle_c\right]^{\dagger} \left[\sum_{N,t} \langle \mathbf{s} \rangle_c \mathbf{y}'\right],$$

where $\mathbf{s}$ and $W$ are the vector and matrix of concatenated $\mathbf{s}_i$ and $W_i$, respectively, $\sum_N$ denotes summation over a data set of $N$ sequences, and $\dagger$ is the Moore-Penrose pseudo-inverse. To estimate the log transition probabilities we solve $\partial Q / \partial [A_i]_{jl} = 0$ subject to the constraint $\sum_j e^{[A_i]_{jl}} = 1$, obtaining

$$[A_i]_{jl}^{\text{new}} = \log \left( \frac{\sum_{N,t} \langle s_{ij}^t s_{il}^{t-1} \rangle_c}{\sum_{N,t,j} \langle s_{ij}^t s_{il}^{t-1} \rangle_c} \right). \tag{4}$$

The covariance matrix can be similarly estimated:

$$C^{\text{new}} = \sum_{N,t} \mathbf{y}\mathbf{y}' - \sum_{N,t} \mathbf{y} \langle \mathbf{s} \rangle_c' \langle \mathbf{ss}' \rangle_c^{\dagger} \langle \mathbf{s} \rangle_c \mathbf{y}'.$$

The M-step equations can therefore be solved analytically; furthermore, for a single underlying Markov chain, they reduce to the traditional Baum-Welch re-estimation equations.

## E-step

Unfortunately, as in the simpler CVQ, the exact E-step for factorial HMMs is computationally intractable. For example, the expectation of the $j^{\text{th}}$ unit in vector $i$ at time step $t$, given $\{\mathbf{y}\}$, is:

$$\langle s_{ij}^t \rangle_c = P(s_{ij}^t = 1 | \{\mathbf{y}\}, \phi)$$

$$= \sum_{j_1, \ldots, j_h \neq i, \ldots, j_d}^{k} P(s_{1j_1}^t = 1, \ldots, s_{ij}^t = 1, \ldots, s_{d,j_d}^t = 1 | \{\mathbf{y}\}, \phi)$$

Although the Markov property can be used to obtain a forward-backward–like factorization of this expectation across time steps, the sum over all possible configurations of the other hidden units *within* each time step is unavoidable. For a data set

of $N$ sequences of length $T$, the full E-step calculated through the forward-backward procedure has time complexity $\mathcal{O}(NTk^{2d})$. Although more careful bookkeeping can reduce the complexity to $\mathcal{O}(NTdk^{d+1})$, the exponential time cannot be avoided. This intractability of the exact E-step is due inherently to the cooperative nature of the model—the setting of one vector only determines the mean of the observable if all the other vectors are fixed.

Rather than summing over all possible hidden state patterns to compute the exact expectations, a natural approach is to approximate them through a Monte Carlo method such as Gibbs sampling. The procedure starts with a clamped observable sequence $\{\mathbf{y}\}$ and a random setting of the hidden states $\{\mathbf{s}_j^t\}$. At each time step, each state vector is updated stochastically according to its probability distribution conditioned on the setting of all the other state vectors: $\mathbf{s}_i^t \sim P(\mathbf{s}_i^t|\{\mathbf{y}\},\{\mathbf{s}_j^\tau : j \neq i \text{ or } \tau \neq t\}, \phi)$. These conditional distributions are straightforward to compute and a full pass of Gibbs sampling requires $\mathcal{O}(NTkd)$ operations. The first and second-order statistics needed to estimate $\langle \mathbf{s}_i^t \rangle_c$, $\langle \mathbf{s}_i^t \mathbf{s}_j^{t'} \rangle_c$ and $\langle \mathbf{s}_i^t \mathbf{s}_i^{t-1'} \rangle_c$ are collected using the $s_{ij}^t$'s visited and the probabilities estimated during this sampling process.

**Mean field approximation**

A different approach to computing the expectations in an intractable system is given by mean field theory. A mean field approximation for factorial HMMs can be obtained by defining the energy function

$$\tilde{\mathcal{H}}(\{\mathbf{s},\mathbf{y}\}) = \frac{1}{2} \sum_t \left[\mathbf{y}^t - \boldsymbol{\mu}^t\right]' C^{-1} \left[\mathbf{y}^t - \boldsymbol{\mu}^t\right] - \sum_{t,i} \mathbf{s}_i^{t'} \log \mathbf{m}_i^t.$$

which results in a completely factorized approximation to probability density (2):

$$\tilde{P}(\{\mathbf{s},\mathbf{y}\}) \propto \prod_t \exp\{-\frac{1}{2} \left[\mathbf{y}^t - \boldsymbol{\mu}^t\right]' C^{-1} \left[\mathbf{y}^t - \boldsymbol{\mu}^t\right]\} \prod_{t,i,j} (\mathbf{m}_{ij}^t)^{s_{ij}^t} \tag{5}$$

In this approximation, the observables are independently Gaussian distributed with mean $\boldsymbol{\mu}^t$ and each hidden state vector is multinomially distributed with mean $\mathbf{m}_i^t$. This approximation is made as tight as possible by chosing the mean field parameters $\boldsymbol{\mu}^t$ and $\mathbf{m}_i^t$ that minimize the Kullback-Liebler divergence

$$\mathcal{KL}(\tilde{P}\|P) \equiv \langle \log P \rangle_{\tilde{P}} - \langle \log \tilde{P} \rangle_{\tilde{P}}$$

where $\langle \cdot \rangle_{\tilde{P}}$ denotes expectation over the mean field distribution (5). With the observables clamped, $\boldsymbol{\mu}^t$ can be set equal to the observable $\mathbf{y}^t$. Minimizing $\mathcal{KL}(\tilde{P}\|P)$ with respect to the mean field parameters for the states results in a fixed-point equation which can be iterated until convergence:

$$\mathbf{m}_i^{t\,\text{new}} = \sigma\{W_i' C^{-1} \left[\mathbf{y}^t - \hat{\mathbf{y}}^t\right] + W_i' C^{-1} W_i \mathbf{m}_i^t - \frac{1}{2}\text{diag}\{W_i' C^{-1} W_i\} - 1 \tag{6}$$
$$+ A_i \mathbf{m}_i^{t-1} + A_i' \mathbf{m}_i^{t+1}\}$$

where $\hat{\mathbf{y}}^t \equiv \sum_i W_i \mathbf{m}_i^t$ and $\sigma\{\cdot\}$ is the softmax exponential, normalized over each hidden state vector. The first term is the projection of the error in the observable onto the weights of state vector $i$—the more a hidden unit can reduce this error, the larger its mean field parameter. The next three terms arise from the fact that $\langle s_{ij}^2 \rangle_{\tilde{P}}$ is equal to $m_{ij}$ and not $m_{ij}^2$. The last two terms introduce dependencies forward and backward in time. Each state vector is asynchronously updated using (6), at a time cost of $\mathcal{O}(NTkd)$ per iteration. Convergence is diagnosed by monitoring the $\mathcal{KL}$ divergence in the mean field distribution between successive time steps; in practice convergence is very rapid (about 2 to 10 iterations of (6)).

Table 1: Comparison of factorial HMM on four problems of varying size

| $d$ | $k$ | Alg | # | Train | | Test | | Cycles | | Time/Cycle |
|---|---|---|---|---|---|---|---|---|---|---|
| 3 | 2 | HMM | 5 | 649 | $\pm$ 8 | 358 | $\pm$ 81 | 33 | $\pm$ 19 | 1.1 s |
| | | Exact | | 877 | $\pm$ 0 | 768 | $\pm$ 0 | 22 | $\pm$ 6 | 3.0 s |
| | | Gibbs | | 710 | $\pm$ 152 | 627 | $\pm$ 129 | 28 | $\pm$ 11 | 6.0 s |
| | | MF | | 755 | $\pm$ 168 | 670 | $\pm$ 137 | 32 | $\pm$ 22 | 1.2 s |
| 3 | 3 | HMM | 5 | 670 | $\pm$ 26 | -782 | $\pm$ 128 | 23 | $\pm$ 10 | 3.6 s |
| | | Exact | | 568 | $\pm$ 164 | 276 | $\pm$ 62 | 35 | $\pm$ 12 | 5.2 s |
| | | Gibbs | | 564 | $\pm$ 160 | 305 | $\pm$ 51 | 45 | $\pm$ 16 | 9.2 s |
| | | MF | | 495 | $\pm$ 83 | 326 | $\pm$ 62 | 38 | $\pm$ 22 | 1.6 s |
| 5 | 2 | HMM | 5 | 588 | $\pm$ 37 | -2634 | $\pm$ 566 | 18 | $\pm$ 1 | 5.2 s |
| | | Exact | | 223 | $\pm$ 76 | 159 | $\pm$ 80 | 31 | $\pm$ 17 | 6.9 s |
| | | Gibbs | | 123 | $\pm$ 103 | 73 | $\pm$ 95 | 40 | $\pm$ 5 | 12.7 s |
| | | MF | | 292 | $\pm$ 101 | 237 | $\pm$ 103 | 54 | $\pm$ 29 | 2.2 s |
| 5 | 3 | HMM | 3 | 1671,1678,1690 | | $-\infty,-\infty,-\infty$ | | 14,14,12 | | 90.0 s |
| | | Exact | | -55,-354,-295 | | -123,-378,-402 | | 90,100,100 | | 51.0 s |
| | | Gibbs | | -123,-160,-194 | | -202,-237,-307 | | 100,73,100 | | 14.2 s |
| | | MF | | -287,-286,-296 | | -364,-370,-365 | | 100,100,100 | | 4.7 s |

Table 1. Data was generated from a factorial HMM with $d$ underlying Markov models of $k$ states each. The training set was 10 sequences of length 20 where the observable was a 4-dimensional vector; the test set was 20 such sequences. HMM indicates a hidden Markov model with $k^d$ states; the other algorithms are factorial HMMs with $d$ underlying $k$-state models. Gibbs sampling used 10 samples of each state. The algorithms were run until convergence, as monitored by relative change in the likelihood, or a maximum of 100 cycles. The # column indicates number of runs. The Train and Test columns show the log likelihood $\pm$ one standard deviation on the two data sets. The last column indicates approximate time per cycle on a Silicon Graphics R4400 processor running Matlab.

## 3 Empirical Results

We compared three EM algorithms for learning in factorial HMMs—using Gibbs sampling, mean field approximation, and the exact (exponential) E step—on the basis of performance and speed on randomly generated problems. Problems were generated from a factorial HMM structure, the parameters of which were sampled from a uniform $[0, 1]$ distribution, and appropriately normalized to satisfy the sum-to-one constraints of the transition matrices and priors. Also included in the comparison was a traditional HMM with as many states ($k^d$) as the factorial HMM.

Table 1 summarizes the results. Even for moderately large state spaces ($d \geq 3$ and $k \geq 3$) the standard HMM with $k^d$ states suffers from severe overfitting. Furthermore, both the standard HMM and the exact E-step factorial HMM are extremely slow on the larger problems. The Gibbs sampling and mean field approximations offer roughly comparable performance at a great increase in speed.

## 4 Discussion

The basic contribution of this paper is a learning algorithm for hidden Markov models with distributed state representations. The standard Baum-Welch procedure is intractable for such architectures as the size of the state space generated from the cross product of $d$ $k$-valued features is $\mathcal{O}(k^d)$, and the time complexity of Baum-Welch is quadratic in this size. More importantly, unless special constraints are applied to this cross-product HMM architecture, the number of parameters also

grows as $\mathcal{O}(k^{2d})$, which can result in severe overfitting.

The architecture for factorial HMMs presented in this paper did not include any coupling between the underlying Markov chains. It is possible to extend the algorithm presented to architectures which incorporate such couplings. However, these couplings must be introduced with caution as they may result either in an exponential growth in parameters or in a loss of the constant partition function property.

The learning algorithm derived in this paper assumed real-valued observables. The algorithm can also be derived for HMMs with discrete observables, an architecture closely related to sigmoid belief networks (Neal, 1992). However, the nonlinearities induced by discrete observables make both the E-step and M-step of the algorithm more difficult.

In conclusion, we have presented Gibbs sampling and mean field learning algorithms for factorial hidden Markov models. Such models incorporate the time series modeling capabilities of hidden Markov models and the advantages of distributed representations for the state space. Future work will concentrate on a more efficient mean field approximation in which the forward-backward algorithm is used to compute the E-step exactly within each Markov chain, and mean field theory is used to handle interactions between chains (Saul and Jordan, 1996).

## Acknowledgements

This project was supported in part by a grant from the McDonnell-Pew Foundation, by a grant from ATR Human Information Processing Research Laboratories, by a grant from Siemens Corporation, and by grant N00014-94-1-0777 from the Office of Naval Research.

## Footnotes

[1] We refer to HMMs with distributed state as *factorial* HMMs as the features of the distributed state factorize the total state representation.

# References

Baldi, P., Chauvin, Y., Hunkapiller, T., and McClure, M. (1994). Hidden Markov models of biological primary sequence information. *Proc. Nat. Acad. Sci. (USA)*, 91(3):1059–1063.

Ghahramani, Z. (1995). Factorial learning and the *EM* algorithm. In Tesauro, G., Touretzky, D., and Leen, T., editors, *Advances in Neural Information Processing Systems 7*. MIT Press, Cambridge, MA.

Hinton, G. and Zemel, R. (1994). Autoencoders, minimum description length, and Helmholtz free energy. In Cowan, J., Tesauro, G., and Alspector, J., editors, *Advances in Neural Information Processing Systems 6*. Morgan Kaufmanm Publishers, San Francisco, CA.

Neal, R. (1992). Connectionist learning of belief networks. *Artificial Intelligence*, 56:71–113.

Rabiner, L. and Juang, B. (1986). An Introduction to hidden Markov models. *IEEE Acoustics, Speech & Signal Processing Magazine*, 3:4–16.

Saul, L. and Jordan, M. (1995). Boltzmann chains and hidden Markov models. In Tesauro, G., Touretzky, D., and Leen, T., editors, *Advances in Neural Information Processing Systems 7*. MIT Press, Cambridge, MA.

Saul, L. and Jordan, M. (1996). Exploiting tractable substructures in Intractable networks. In Touretzky, D., Mozer, M., and Hasselmo, M., editors, *Advances in Neural Information Processing Systems 8*. MIT Press.

Williams, C. and Hinton, G. (1991). Mean field networks that learn to discriminate temporally distorted strings. In Touretzky, D., Elman, J., Sejnowski, T., and Hinton, G., editors, *Connectionist Models: Proceedings of the 1990 Summer School*, pages 18–22. Morgan Kaufmann Publishers, Man Mateo, CA.